# Nonlinear causal discovery with additive noise models

**Patrik O. Hoyer**
University of Helsinki
Finland

**Dominik Janzing**
MPI for Biological Cybernetics
Tübingen, Germany

**Joris Mooij**
MPI for Biological Cybernetics
Tübingen, Germany

**Jonas Peters**
MPI for Biological Cybernetics
Tübingen, Germany

**Bernhard Schölkopf**
MPI for Biological Cybernetics
Tübingen, Germany

## Abstract

The discovery of causal relationships between a set of observed variables is a fundamental problem in science. For continuous-valued data *linear* acyclic causal models with additive noise are often used because these models are well understood and there are well-known methods to fit them to data. In reality, of course, many causal relationships are more or less *nonlinear*, raising some doubts as to the applicability and usefulness of purely linear methods. In this contribution we show that the basic linear framework can be generalized to nonlinear models. In this extended framework, nonlinearities in the data-generating process are in fact a blessing rather than a curse, as they typically provide information on the underlying causal system and allow more aspects of the true data-generating mechanisms to be identified. In addition to theoretical results we show simulations and some simple real data experiments illustrating the identification power provided by nonlinearities.

## 1 Introduction

Causal relationships are fundamental to science because they enable predictions of the consequences of actions [1]. While controlled randomized experiments constitute the primary tool for identifying causal relationships, such experiments are in many cases either unethical, too expensive, or technically impossible. The development of *causal discovery* methods to infer causal relationships from uncontrolled data constitutes an important current research topic [1, 2, 3, 4, 5, 6, 7, 8]. If the observed data is continuous-valued, methods based on *linear* causal models (aka structural equation models) are commonly applied [1, 2, 9]. This is not necessarily because the true causal relationships are really believed to be linear, but rather it reflects the fact that linear models are well understood and easy to work with. A standard approach is to estimate a so-called *Markov equivalence class* of directed acyclic graphs (all representing the same conditional independencies) from the data [1, 2, 3]. For continuous variables, the independence tests often assume linear models with additive Gaussian noise [2]. Recently, however, it has been shown that for linear models, *non-Gaussianity* in the data can actually aid in distinguishing the causal directions and allow one to uniquely identify the generating graph under favourable conditions [7]. Thus the practical case of non-Gaussian data which long was considered a nuisance turned out to be helpful in the causal discovery setting.

In this contribution we show that nonlinearities can play a role quite similar to that of non-Gaussianity: When causal relationships are nonlinear it typically helps break the symmetry between the observed variables and allows the identification of causal directions. As Friedman and Nachman have pointed out [10], non-invertible functional relationships between the observed variables can provide clues to the generating causal model. However, we show that the phenomenon is much more general; for nonlinear models with additive noise *almost any* nonlinearities (invertible or not) will typically yield identifiable models. Note that other methods to select among Markov equivalent DAGs [11, 8] have (so far) mainly focussed on mixtures of discrete and continuous variables.

In the next section, we start by defining the family of models under study, and then, in Section 3 we give theoretical results on the identifiability of these models from non-interventional data. We describe a practical method for inferring the generating model from a sample of data vectors in Section 4, and show its utility in simulations and on real data (Section 5).

## 2 Model definition

We assume that the observed data has been generated in the following way: Each observed variable $x_i$ is associated with a node $i$ in a directed acyclic graph $G$, and the value of $x_i$ is obtained as a function of its parents in $G$, plus independent additive noise $n_i$, i.e.

$$x_i := f_i(\mathbf{x}_{\mathbf{pa}(i)}) + n_i \qquad (1)$$

where $f_i$ is an arbitrary function (possibly different for each $i$), $\mathbf{x}_{\mathbf{pa}(i)}$ is a vector containing the elements $x_j$ such that there is an edge from $j$ to $i$ in the DAG $G$, the noise variables $n_i$ may have arbitrary probability densities $p_{n_i}(n_i)$, and the noise variables are jointly independent, that is $p_{\mathbf{n}}(\mathbf{n}) = \prod_i p_{n_i}(n_i)$, where $\mathbf{n}$ denotes the vector containing the noise variables $n_i$. Our data then consists of a number of vectors $\mathbf{x}$ sampled independently, each using $G$, the same functions $f_i$, and the $n_i$ sampled independently from the same densities $p_{n_i}(n_i)$.

Note that this model includes the special case when all the $f_i$ are linear and all the $p_{n_i}$ are Gaussian, yielding the standard linear–Gaussian model family [2, 3, 9]. When the functions are linear but the densities $p_{n_i}$ are non-Gaussian we obtain the linear–non-Gaussian models described in [7].

The goal of causal discovery is, given the data vectors, to infer as much as possible about the generating mechanism; in particular, we seek to infer the generating graph $G$. In the next section we discuss the prospects of this task in the theoretical case where the joint distribution $p_{\mathbf{x}}(\mathbf{x})$ of the observed data can be estimated exactly. Later, in Section 4, we experimentally tackle the practical case of a finite-size data sample.

## 3 Identifiability

Our main theoretical results concern the simplest non-trivial graph: the case of two variables. The experimental results will, however, demonstrate that the basic principle works even in the general case of $N$ variables.

Figure 1 illustrates the basic identifiability principle for the two-variable model. Denoting the two variables $x$ and $y$, we are considering the generative model $y := f(x) + n$ where $x$ and $n$ are

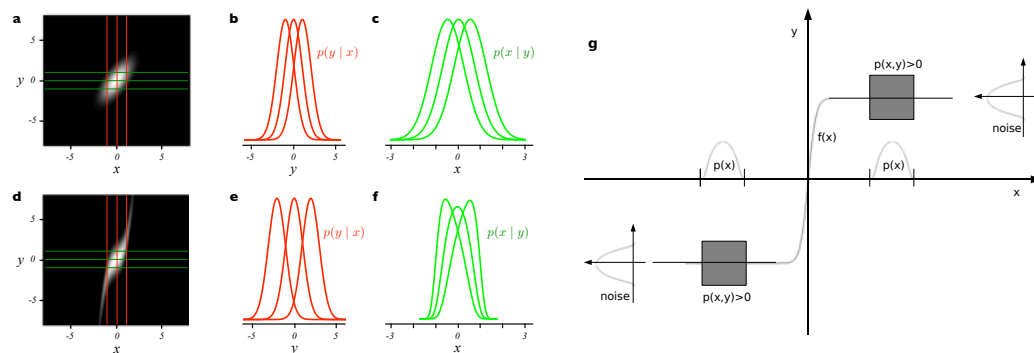

Figure 1: Identification of causal direction based on constancy of conditionals. See main text for a detailed explanation of (a)–(f). (g) shows an example of a joint density $p(x\ y)$ generated by a causal model $x \to y$ with $y := f(x) + n$ where $f$ is nonlinear, the supports of the densities $p_x(x)$ and $p_n(n)$ are compact regions, and the function $f$ is constant on each connected component of the support of $p_x$. The support of the joint density is now given by the two gray squares. Note that the input distribution $p_x$, the noise distribution $p_n$ and $f$ can in fact be chosen such that the joint density is symmetrical with respect to the two variables, i.e. $p(x\ y) = p(y\ x)$, making it obvious that there will also be a valid backward model.

both Gaussian and statistically independent. In panel **(a)** we plot the joint density $p(x, y)$ of the observed variables, for the linear case of $f(x) = x$. As a trivial consequence of the model, the conditional density $p(y \mid x)$ has identical shape for all values of $x$ and is simply shifted by the function $f(x)$; this is illustrated in panel **(b)**. In general, there is no reason to believe that this relationship would also hold for the conditionals $p(x \mid y)$ for different values of $y$ but, as is well known, for the linear–Gaussian model this is actually the case, as illustrated in panel **(c)**. Panels **(d-f)** show the corresponding joint and conditional densities for the corresponding model with a nonlinear function $f(x) = x + x^3$. Notice how the conditionals $p(x \mid y)$ look different for different values of $y$, indicating that a reverse causal model of the form $x := g(y) + \tilde{n}$ (with $y$ and $\tilde{n}$ statistically independent) would not be able to fit the joint density. As we will show in this section, this will in fact *typically* be the case, however, not always.

To see the latter, we first show that there exist models other than the linear–Gaussian and the independent case which admit both a forward $x \rightarrow y$ and a backward $x \leftarrow y$ model. Panel **(g)** of Figure 1 presents a nonlinear functional model with additive non-Gaussian noise and non-Gaussian input distributions that nevertheless admits a backward model. The functions and probability densities can be chosen to be (arbitrarily many times) differentiable. Note that the example of panel **(g)** in Figure 1 is somewhat artificial: $p$ has compact support, and $x, y$ are independent inside the connected components of the support. Roughly speaking, the nonlinearity of $f$ does not matter since it occurs where $p$ is zero — an artifical situation which is avoided by the requirement that from now on, we will assume that all probability densities are strictly positive. Moreover, we assume that all functions (including densities) are three times differentiable. In this case, the following theorem shows that for generic choices of $f$, $p_x(x)$, and $p_n(n)$, there exists no backward model.

**Theorem 1** *Let the joint probability density of $x$ and $y$ be given by*

$$p(x, y) = p_n(y - f(x))p_x(x),\tag{2}$$

*where $p_n, p_x$ are probability densities on $\mathbb{R}$. If there is a backward model of the same form, i.e.,*

$$p(x, y) = p_{\tilde{n}}(x - g(y))p_y(y),\tag{3}$$

*then, denoting $\nu := \log p_n$ and $\xi := \log p_x$, the triple $(f, p_x, p_n)$ must satisfy the following differential equation for all $x, y$ with $\nu''(y - f(x))f'(x) \neq 0$:*

$$\xi''' = \xi'' \left( -\frac{\nu''' f'}{\nu''} + \frac{f''}{f'} \right) - 2\nu'' f'' f' + \nu' f''' + \frac{\nu' \nu''' f'' f'}{\nu''} - \frac{\nu'(f'')^2}{f'},\tag{4}$$

*where we have skipped the arguments $y - f(x)$, $x$, and $x$ for $\nu$, $\xi$, and $f$ and their derivatives, respectively. Moreover, if for a fixed pair $(f, \nu)$ there exists $y \in \mathbb{R}$ such that $\nu''(y - f(x))f'(x) \neq 0$ for all but a countable set of points $x \in \mathbb{R}$, the set of all $p_x$ for which $p$ has a backward model is contained in a 3-dimensional affine space.*

Loosely speaking, the statement that the differential equation for $\xi$ has a 3-dimensional space of solutions (while a priori, the space of all possible log-marginals $\xi$ is infinite dimensional) amounts to saying that in the generic case, our forward model cannot be inverted.

A simple corollary is that if both the marginal density $p_x(x)$ and the noise density $p_n(y - f(x))$ are Gaussian then the existence of a backward model implies linearity of $f$:

**Corollary 1** *Assume that $\nu''' = \xi''' = 0$ everywhere. If a backward model exists, then $f$ is linear.*

The proofs of Theorem 1 and Corollary 1 are provided in the Appendix.

Finally, we note that even when $f$ is linear and $p_n$ and $p_x$ are non-Gaussian, although a *linear* backward model has previously been ruled out [7], there exist special cases where there is a *nonlinear* backward model with independent additive noise. One such case is when $f(x) = -x$ and $p_x$ and $p_n$ are Gumbel distributions: $p_x(x) = \exp(-x - \exp(-x))$ and $p_n(n) = \exp(-n - \exp(-n))$. Then taking $p_y(y) = \exp(-y - 2\log(1 + \exp(-y)))$, $p_{\tilde{n}}(\tilde{n}) = \exp(-2\tilde{n} - \exp(-\tilde{n}))$ and $g(y) = \log(1 + \exp(-y))$ one obtains $p(x, y) = p_n(y - f(x))p_x(x) = p_{\tilde{n}}(x - g(y))p_y(y)$.

Although the above results strictly only concern the two-variable case, there are strong reasons to believe that the general argument also holds for larger models. In this brief contribution we do not pursue any further theoretical results, rather we show empirically that the estimation principle can be extended to networks involving more than two variables.

# 4 Model estimation

Section 3 established for the two-variable case that given knowledge of the exact densities, the true model is (in the generic case) identifiable. We now consider practical estimation methods which infer the generating graph from sample data.

Again, we begin by considering the case of two observed scalar variables $x$ and $y$. Our basic method is straightforward: First, test whether $x$ and $y$ are statistically independent. If they are not, we continue as described in the following manner: We test whether a model $y := f(x) + n$ is consistent with the data, simply by doing a nonlinear regression of $y$ on $x$ (to get an estimate $\hat{f}$ of $f$), calculating the corresponding residuals $\hat{n} = y - \hat{f}(x)$, and testing whether $\hat{n}$ is independent of $x$. If so, we accept the model $y := f(x) + n$; if not, we reject it. We then similarly test whether the reverse model $x := g(y) + n$ fits the data.

The above procedure will result in one of several possible scenarios. First, if $x$ and $y$ are deemed mutually independent we infer that there is no causal relationship between the two, and no further analysis is performed. On the other hand, if they are dependent but both directional models are accepted we conclude that either model may be correct but we cannot infer it from the data. A more positive result is when we are able to reject one of the directions and (tentatively) accept the other. Finally, it may be the case that neither direction is consistent with the data, in which case we conclude that the generating mechanism is more complex and cannot be described using this model.

This procedure could be generalized to an arbitrary number $N$ of observed variables, in the following way: For each DAG $G_i$ over the observed variables, test whether it is consistent with the data by constructing a nonlinear regression of each variable on its parents, and subsequently testing whether the resulting residuals are mutually independent. If any independence test is rejected, $G_i$ is rejected. On the other hand, if none of the independence tests are rejected, $G_i$ is consistent with the data.

The above procedure is obviously feasible only for very small networks (roughly $N \leq 7$ or so) and also suffers from the problem of multiple hypothesis testing; an important future improvement would be to take this properly into account. Furthermore, the above algorithm returns *all* DAGs consistent with the data, including all those for which consistent subgraphs exist. Our current implementation removes any such unnecessarily complex graphs from the output.

The selection of the nonlinear regressor and of the particular independence tests are not constrained. Any prior information on the types of functional relationships or the distributions of the noise should optimally be utilized here. In our implementation, we perform the regression using Gaussian Processes [12] and the independence tests using kernel methods [13]. Note that one must take care to avoid overfitting, as overfitting may lead one to falsely accept models which should be rejected.

# 5 Experiments

To show the ability of our method to find the correct model when all the assumptions hold we have applied our implementation to a variety of simulated and real data.

For the regression, we used the GPML code from [14] corresponding to [12], using a Gaussian kernel and independent Gaussian noise, optimizing the hyperparameters for each regression individually.[1] In principle, any regression method can be used; we have verified that our results do not depend significantly on the choice of the regression method by comparing with $\nu$-SVR [15] and with thin-plate spline kernel regression [16]. For the independence test, we implemented the HSIC [13] with a Gaussian kernel, where we used the gamma distribution as an approximation for the distribution of the HSIC under the null hypothesis of independence in order to calculate the $p$-value of the test result.

**Simulations.** The main results for the two-variable case are shown in Figure 2. We simulated data using the model $y = x + bx^3 + n$; the random variables $x$ and $n$ were sampled from a Gaussian distribution and their absolute values were raised to the power $q$ while keeping the original sign.

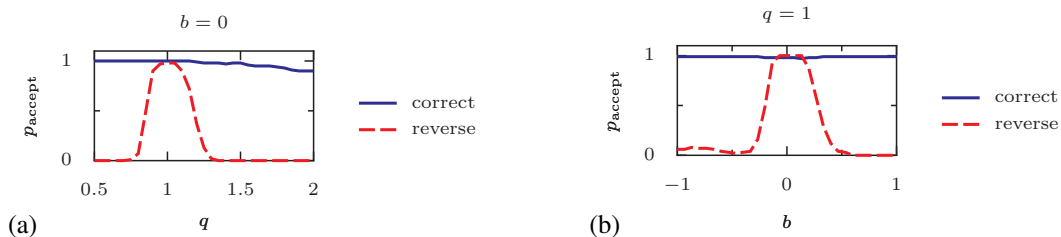

Figure 2: Results of simulations (see main text for details): **(a)** The proportion of times the forward and the reverse model were accepted, $p_{\text{accept}}$, as a function of the non-Gaussianity parameter $q$ (for $b = 0$), and **(b)** as a function of the nonlinearity parameter $b$ (for $q = 1$).

The parameter $b$ controls the strength of the nonlinearity of the function, $b = 0$ corresponding to the linear case. The parameter $q$ controls the non-Gaussianity of the noise: $q = 1$ gives a Gaussian, while $q > 1$ and $q < 1$ produces super-Gaussian and sub-Gaussian distributions respectively. We used 300 $(x, y)$ samples for each trial and used a significance level of 2% for rejecting the null hypothesis of independence of residuals and cause. For each $b$ value (or $q$ value) we repeated the experiment 100 times in order to estimate the acceptance probabilities. Panel **(a)** shows that our method can solve the well-known linear but non-Gaussian special case [7]. By plotting the acceptance probability of the correct and the reverse model as a function of non-Gaussianity we can see that when the distributions are sufficiently non-Gaussian the method is able to infer the correct causal direction. Then, in panel **(b)** we similarly demonstrate that we can identify the correct direction for the Gaussian marginal and Gaussian noise model when the functional relationship is sufficiently nonlinear. Note in particular, that the model is identifiable also for positive $b$ in which case the function is invertible, indicating that non-invertibility is not a necessary condition for identification.

We also did experiments for 4 variables $w, x, y$ and $z$ with a diamond-like causal structure. We took $w \sim U(-3, 3)$, $x = w^2 + n_x$ with $n_x \sim U(-1, 1)$, $y = 4\sqrt{|w|} + n_y$ with $n_y \sim U(-1, 1)$, $z = 2\sin x + 2\sin y + n_z$ with $n_z \sim U(-1, 1)$. We sampled 500 $(w, x, y, z)$ tuples from the model and applied the algorithm described in Section 4 in order to reconstruct the DAG structure. The simplest DAG that was consistent with the data (with significance level 2% for each test) turned out to be precisely the true causal structure. All five other DAGs for which the true DAG is a subgraph were also consistent with the data.

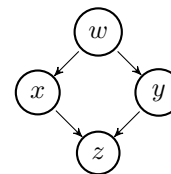

**Real-world data.** Of particular empirical interest is how well the proposed method performs on real world datasets for which the assumptions of our method might only hold approximately. Due to space constraints we only discuss three real world datasets here.

The first dataset, the "Old Faithful" dataset [17] contains data about the duration of an eruption and the time interval between subsequent eruptions of the Old Faithful geyser in Yellowstone National Park, USA. Our method obtains a $p$-value of 0.5 for the (forward) model "current duration causes next interval length" and a $p$-value of $4.4 \times 10^{-9}$ for the (backward) model "next interval length causes current duration". Thus, we accept the model where the time interval between the current and the next eruption is a function of the duration of the current eruption, but reject the reverse model. This is in line with the chronological ordering of these events. Figure 3 illustrates the data, the forward and backward fit and the residuals for both fits. Note that for the forward model, the residuals seem to be independent of the duration, whereas for the backward model, the residuals are clearly dependent on the interval length. Time-shifting the data by one time step, we obtain for the (forward) model "current interval length causes next duration" a $p$-value smaller than $10^{-15}$ and for the (backward) model "next duration causes current interval length" we get a $p$-value of $1.8 \times 10^{-8}$. Hence, our simple nonlinear model with independent additive noise is not consistent with the data in either direction.

The second dataset, the "Abalone" dataset from the UCI ML repository [18], contains measurements of the number of rings in the shell of abalone (a group of shellfish), which indicate their age, and the length of the shell. Figure 4 shows the results for a subsample of 500 datapoints. The correct model "age causes length" leads to a $p$-value of 0.19, while the reverse model "length causes age" comes

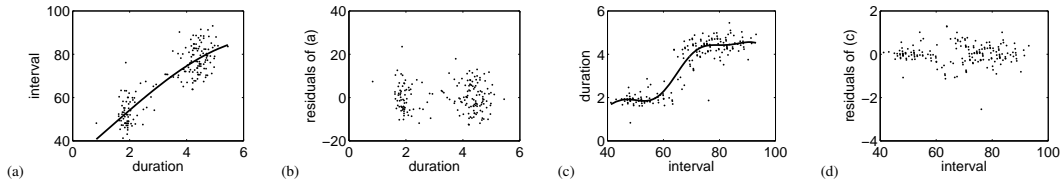

Figure 3: The Old Faithful Geyser data: (a) forward fit corresponding to "current duration causes next interval length"; (b) residuals for forward fit; (c) backward fit corresponding to "next interval length causes current duration"; (d) residuals for backward fit.

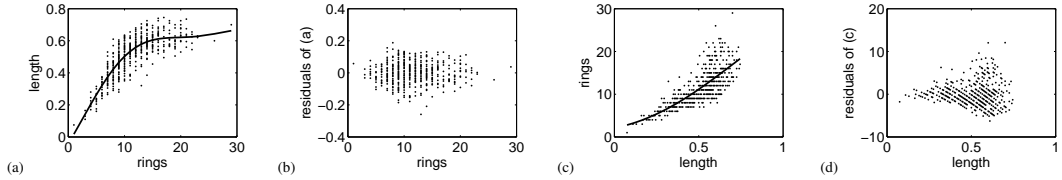

Figure 4: Abalone data: (a) forward fit corresponding to "age (rings) causes length"; (b) residuals for forward fit; (c) backward fit corresponding to "length causes age (rings)"; (d) residuals for backward fit.

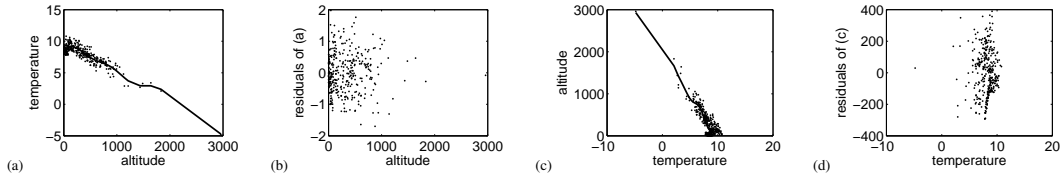

Figure 5: Altitude–temperature data. (a) forward fit corresponding to "altitude causes temperature"; (b) residuals for forward fit; (c) backward fit corresponding to "temperature causes altitude"; (d) residuals for backward fit.

with $p < 10^{-15}$. This is in accordance with our intuition. Note that our method favors the correct direction although the assumption of independent additive noise is only approximately correct here; indeed, the variance of the length is dependent on age.

Finally, we assay the method on a simple example involving two observed variables: The altitude above sea level (in meters) and the local yearly average outdoor temperature in centigrade, for 349 weather stations in Germany, collected over the time period of 1961–1990 [19]. The correct model "altitude causes temperature" leads to $p = 0.017$, while "temperature causes altitude" can clearly be rejected ($p = 8 \times 10^{-15}$), in agreement with common understanding of causality in this case. The results are shown in Figure 5.

## 6  Conclusions

In this paper, we have shown that the linear–non-Gaussian causal discovery framework can be generalized to admit nonlinear functional dependencies as long as the noise on the variables remains additive. In this approach nonlinear relationships are in fact helpful rather than a hindrance, as they tend to break the symmetry between the variables and allow the correct causal directions to be identified. Although there exist special cases which admit reverse models we have shown that in the generic case the model is identifiable. We have illustrated our method on both simulated and real world datasets.

## Acknowledgments

We thank Kun Zhang for pointing out an error in the original manuscript. This work was supported in part by the IST Programme of the European Community, under the PASCAL2 Network of Excellence, IST-2007-216886. P.O.H. was supported by the Academy of Finland and by University of Helsinki Research Funds.

## A  Proof of Theorem 1

Set

$$\pi(x, y) := \log p(x, y) = \nu(y - f(x)) + \xi(x) \,, \tag{5}$$

and $\tilde{\nu} := \log p_{\tilde{n}}$, $\eta := \log p_y$. If eq. (3) holds, then $\pi(x, y) = \tilde{\nu}(x - g(y)) + \eta(y)$, implying

$$\frac{\partial^2 \pi}{\partial x \partial y} = -\tilde{\nu}''(x - g(y))g'(y) \quad \text{and} \quad \frac{\partial^2 \pi}{\partial x^2} = \tilde{\nu}''(x - g(y)) \,.$$

We conclude

$$\frac{\partial}{\partial x} \left( \frac{\partial^2 \pi / \partial x^2}{\partial^2 \pi / (\partial x \partial y)} \right) = 0 \,. \tag{6}$$

Using eq. (5) we obtain

$$\frac{\partial^2 \pi}{\partial x \partial y} = -\nu''(y - f(x))f'(x) \,, \tag{7}$$

and

$$\frac{\partial^2 \pi}{\partial x^2} = \frac{\partial}{\partial x} \left( -\nu'(y - f(x))f'(x) + \xi'(x) \right) = \nu''(f')^2 - \nu' f'' + \xi'' \,, \tag{8}$$

where we have dropped the arguments for convenience. Combining eqs. (7) and (8) yields

$$\frac{\partial}{\partial x} \left( \frac{\frac{\partial^2 \pi}{\partial x^2}}{\frac{\partial^2 \pi}{\partial x \partial y}} \right) = -2f'' + \frac{\nu' f'''}{\nu'' f'} - \xi''' \frac{1}{\nu'' f'} + \frac{\nu' \nu''' f''}{(\nu'')^2} - \frac{\nu'(f'')^2}{\nu''(f')^2} - \xi'' \frac{\nu'''}{(\nu'')^2} + \xi'' \frac{f''}{\nu''(f')^2} \,.$$

Due to eq. (6) this expression must vanish and we obtain DE (4) by term reordering. Given $f, \nu$, we obtain for every fixed $y$ a linear inhomogeneous DE for $\xi$:

$$\xi'''(x) = \xi''(x)G(x, y) + H(x, y) \,, \tag{9}$$

where $G$ and $H$ are defined by

$$G := -\frac{\nu''' f'}{\nu''} + \frac{f''}{f'} \quad \text{and} \quad H := -2\nu'' f'' f' + \nu' f''' + \frac{\nu' \nu''' f'' f'}{\nu''} - \frac{\nu'(f'')^2}{f'} \,.$$

Setting $z := \xi''$ we have $z'(x) = z(x)G(x, y) + H(x, y)$. Given that such a function $z$ exists, it is given by

$$z(x) = z(x_0)e^{\int_{x_0}^{x} G(\tilde{x}, y)d\tilde{x}} + \int_{x_0}^{x} e^{\int_{\hat{x}}^{x} G(\tilde{x}, y)d\tilde{x}} H(\hat{x}, y)d\hat{x} \,. \tag{10}$$

Let $y$ be fixed such that $\nu''(y - f(x))f'(x) \neq 0$ holds for all but countably many $x$. Then $z$ is determined by $z(x_0)$ since we can extend eq. (10) to the remaining points. The set of all functions $\xi$ satisfying the linear inhomogenous DE (9) is a 3-dimensional affine space: Once we have fixed $\xi(x_0), \xi'(x_0), \xi''(x_0)$ for some arbitrary point $x_0$, $\xi$ is completely determined. Given fixed $f$ and $\nu$, the set of all $\xi$ admitting a backward model is contained in this subspace. $\qquad\square$

## B  Proof of Corollary 1

Similarly to how (6) was derived, under the assumption of the existence of a reverse model one can derive

$$\frac{\partial^2 \pi}{\partial x \partial y} \cdot \frac{\partial}{\partial x} \left( \frac{\partial^2 \pi}{\partial x^2} \right) = \frac{\partial^2 \pi}{\partial x^2} \cdot \frac{\partial}{\partial x} \left( \frac{\partial^2 \pi}{\partial x \partial y} \right)$$

Now using (7) and (8), we obtain

$$(-\nu'' f') \cdot \frac{\partial}{\partial x} \left( \nu'' (f')^2 - \nu' f'' + \xi'' \right) = \left( \nu'' (f')^2 - \nu' f'' + \xi'' \right) \cdot \frac{\partial}{\partial x} \left( -\nu'' f' \right)$$

which reduces to

$$-2(\nu'' f')^2 f'' + \nu'' f' \nu' f''' - \nu'' f' \xi''' = -\nu' f'' \nu''' (f')^2 + \xi'' \nu''' (f')^2 + \nu'' \nu' (f'')^2 - \nu'' f'' \xi''.$$

Substituting the assumptions $\xi''' = 0$ and $\nu''' = 0$ (and hence $\nu'' = C$ everywhere with $C \neq 0$ since otherwise $\nu$ cannot be a proper log-density) yields

$$\nu' \big( y - f(x) \big) \cdot \big( f' f''' - (f'')^2 \big) = 2C(f')^2 f'' - f'' \xi''.$$

Since $C \neq 0$ there exists an $\alpha$ such that $\nu'(\alpha) = 0$. Then, restricting ourselves to the submanifold $\{(x,y) \in \mathbb{R}^2 : y - f(x) = \alpha\}$ on which $\nu' = 0$, we have

$$0 = f''(2C(f')^2 - \xi'').$$

Therefore, for all $x$ in the open set $[f'' \neq 0]$, we have $(f'(x))^2 = \xi''/(2C)$ which is a constant, so $f'' = 0$ on $[f'' \neq 0]$: a contradiction. Therefore, $f'' = 0$ everywhere. □

## Footnotes

[1]The assumption of Gaussian noise is somewhat inconsistent with our general setting where the residuals are allowed to have any distribution (we even prefer the noise to be non-Gaussian); in practice however, the regression yields acceptable results as long as the noise is sufficiently similar to Gaussian noise. In case of significant outliers, other regression methods may yield better results.

### References

[1] J. Pearl. *Causality: Models, Reasoning, and Inference*. Cambridge University Press, 2000.

[2] P. Spirtes, C. Glymour, and R. Scheines. *Causation, Prediction, and Search*. Springer-Verlag, 1993. (2nd ed. MIT Press 2000).

[3] D. Geiger and D. Heckerman. Learning Gaussian networks. In *Proc. of the 10th Annual Conference on Uncertainty in Artificial Intelligence*, pages 235–243, 1994.

[4] D. Heckerman, C. Meek, and G. Cooper. A Bayesian approach to causal discovery. In C. Glymour and G. F. Cooper, editors, *Computation, Causation, and Discovery*, pages 141–166. MIT Press, 1999.

[5] T. Richardson and P. Spirtes. Automated discovery of linear feedback models. In C. Glymour and G. F. Cooper, editors, *Computation, Causation, and Discovery*, pages 253–304. MIT Press, 1999.

[6] R. Silva, R. Scheines, C. Glymour, and P. Spirtes. Learning the structure of linear latent variable models. *Journal of Machine Learning Research*, 7:191–246, 2006.

[7] S. Shimizu, P. O. Hoyer, A. Hyvärinen, and A. J. Kerminen. A linear non-Gaussian acyclic model for causal discovery. *Journal of Machine Learning Research*, 7:2003–2030, 2006.

[8] X. Sun, D. Janzing, and B. Schölkopf. Distinguishing between cause and effect via kernel-based complexity measures for conditional probability densities. *Neurocomputing*, pages 1248–1256, 2008.

[9] K. A. Bollen. *Structural Equations with Latent Variables*. John Wiley & Sons, 1989.

[10] N. Friedman and I. Nachman. Gaussian process networks. In *Proc. of the 16th Annual Conference on Uncertainty in Artificial Intelligence*, pages 211–219, 2000.

[11] X. Sun, D. Janzing, and B. Schölkopf. Causal inference by choosing graphs with most plausible Markov kernels. In *Proceeding of the 9th Int. Symp. Art. Int. and Math.*, Fort Lauderdale, Florida, 2006.

[12] C. E. Rasmussen and C. Williams. *Gaussian Processes for Machine Learning*. MIT Press, 2006.

[13] A. Gretton, R. Herbrich, A. Smola, O. Bousquet, and B. Schölkopf. Kernel methods for measuring independence. *Journal of Machine Learning Research*, 6:2075–2129, 2005.

[14] GPML code. http://www.gaussianprocess.org/gpml/code.

[15] B. Schölkopf, A. J. Smola, and R. Williamson. Shrinking the tube: A new support vector regression algorithm. In *Advances in Neural Information Processing 11 (Proc. NIPS*1998)*. MIT Press, 1999.

[16] G. Wahba. *Spline Models for Observational Data*. Series in Applied Math., Vol. 59, SIAM, Philadelphia, 1990.

[17] A. Azzalini and A. W. Bowman. A look at some data on the Old Faithful Geyser. *Applied Statistics*, 39(3):357–365, 1990.

[18] A. Asuncion and D.J. Newman. UCI machine learning repository, 2007.

[19] Climate data collected by the Deutscher Wetter Dienst. http://www.dwd.de/.
